# On the Concentration of Spectral Properties

**John Shawe-Taylor**
Royal Holloway, University of London
*john@cs.rhul.ac.uk*

**Nello Cristianini**
BIOwulf Technologies
*nello@support-vector.net*

**Jaz Kandola**
Royal Holloway, University of London
*jaz@cs.rhul.ac.uk*

## Abstract

We consider the problem of measuring the eigenvalues of a randomly drawn sample of points. We show that these values can be reliably estimated as can the sum of the tail of eigenvalues. Furthermore, the residuals when data is projected into a subspace is shown to be reliably estimated on a random sample. Experiments are presented that confirm the theoretical results.

## 1 Introduction

A number of learning algorithms rely on estimating spectral data on a sample of training points and using this data as input to further analyses. For example in Principal Component Analysis (PCA) the subspace spanned by the first $k$ eigenvectors is used to give a $k$ dimensional model of the data with minimal residual, hence forming a low dimensional representation of the data for analysis or clustering. Recently the approach has been applied in kernel defined feature spaces in what has become known as kernel-PCA [5]. This representation has also been related to an Information Retrieval algorithm known as latent semantic indexing, again with kernel defined feature spaces [2].

Furthermore eigenvectors have been used in the HITS [3] and Google's PageRank [1] algorithms. In both cases the entries in the eigenvector corresponding to the maximal eigenvalue are interpreted as authority weightings for individual articles or web pages.

The use of these techniques raises the question of how reliably these quantities can be estimated from a random sample of data, or phrased differently, how much data is required to obtain an accurate empirical estimate with high confidence. Ng *et al.* [6] have undertaken a study of the sensitivity of the estimate of the first eigenvector to perturbations of the connection matrix. They have also highlighted the potential instability that can arise when two eigenvalues are very close in value, so that their eigenspaces become very difficult to distinguish empirically.

The aim of this paper is to study the error in estimation that can arise from the random sampling rather than from perturbations of the connectivity. We address

this question using concentration inequalities. We will show that eigenvalues estimated from a sample of size $m$ are indeed concentrated, and furthermore the sum of the last $m - k$ eigenvalues is subject to a similar concentration effect, both results of independent mathematical interest. The sum of the last $m - k$ eigenvalues is related to the error in forming a $k$ dimensional PCA approximation, and hence will be shown to justify using empirical projection subspaces in such algorithms as kernel-PCA and latent semantic kernels.

The paper is organised as follows. In section 2 we give the background results and develop the basic techniques that are required to derive the main results in section 3. We provide experimental verification of the theoretical findings in section 4, before drawing our conclusions.

## 2 Background and Techniques

We will make use of the following results due to McDiarmid. Note that $\mathbb{E}_S$ is the expectation operator under the selection of the sample.

**Theorem 1** *(McDiarmid [4]) Let $X_1, \ldots, X_n$ be independent random variables taking values in a set $A$, and assume that $f : A^n \to \mathbb{R}$, and $f_i : A^{n-1} \to \mathbb{R}$ satisfy for $1 \le i \le n$*

$$\sup_{x_1, \ldots, x_n} |f(x_1, \ldots, x_n) - f_i(x_1, \ldots, x_{i-1}, x_{i+1}, \ldots, x_n)| \le c_i,$$

*then for all $\epsilon > 0$,*

$$P\{|f(X_1, \ldots, X_n) - \mathbb{E}f(X_1, \ldots, X_n)| > \epsilon\} \le 2\exp\left(\frac{-2\epsilon^2}{\sum_{i=1}^n c_i^2}\right)$$

**Theorem 2** *(McDiarmid [4]) Let $X_1, \ldots, X_n$ be independent random variables taking values in a set $A$, and assume that $f : A^n \to \mathbb{R}$, for $1 \le i \le n$*

$$\sup_{x_1, \ldots, x_n, \hat{x}_i} |f(x_1, \ldots, x_n) - f(x_1, \ldots, x_{i-1}, \hat{x}_i, x_{i+1}, \ldots, x_n)| \le c_i,$$

*then for all $\epsilon > 0$,*

$$P\{|f(X_1, \ldots, X_n) - \mathbb{E}f(X_1, \ldots, X_n)| > \epsilon\} \le 2\exp\left(\frac{-2\epsilon^2}{\sum_{i=1}^n c_i^2}\right)$$

We will also make use of the following theorem characterising the eigenvectors of a symmetric matrix.

**Theorem 3 (Courant-Fischer Minimax Theorem)** *If $M \in \mathbb{R}^{m \times m}$ is symmetric, then for $k = 1, \ldots, m$,*

$$\lambda_k(M) = \max_{\dim(T)=k} \min_{0 \ne v \in T} \frac{v'Mv}{v'v} = \min_{\dim(T)=m-k+1} \max_{0 \ne v \in T} \frac{v'Mv}{v'v},$$

*with the extrama achieved by the corresponding eigenvector.*

The approach adopted in the proofs of the next section is to view the eigenvalues as the sums of squares of residuals. This is applicable when the matrix is positive semi-definite and hence can be written as an inner product matrix $M = X'X$, where $X'$ is the transpose of the matrix $X$ containing the $m$ vectors $x_1, \ldots, x_m$ as columns. This is the finite dimensional version of Mercer's theorem, and follows immediately if we take $X = V\sqrt{\Lambda}$, where $M = V\Lambda V'$ is the eigenvalue decomposition of $M$. There may be more succinct ways of representing $X$, but we will assume for simplicity (but without loss of generality) that $X$ is a square matrix with the same dimensions as $M$. To set the scene, we now present a short description of the residuals viewpoint.

The starting point is the singular value decomposition of $X = U\Sigma V'$, where $U$ and $V$ are orthonormal matrices and $\Sigma$ is a diagonal matrix containing the singular values (in descending order). We can now reconstruct the eigenvalue decomposition of $M = X'X = V\Sigma U'U\Sigma V' = V\Lambda V'$, where $\Lambda = \Sigma^2$. But equally we can construct a matrix $N = XX' = U\Sigma V'V\Sigma U' = U\Lambda U'$, with the same eigenvalues as $M$.

As a simple example consider now the first eigenvalue, which by Theorem 3 and the above observations is given by

$$
\begin{aligned}
\lambda_1(M) &= \max_{0 \neq v \in \mathbb{R}^m} \frac{v'Nv}{v'v} = \max_{0 \neq v \in \mathbb{R}^m} \frac{v'XX'v}{v'v} = \max_{0 \neq v \in \mathbb{R}^m} \frac{\|v'X\|^2}{v'v} \\
&= \max_{0 \neq v \in \mathbb{R}^m} \sum_{j=1}^{m} \|P_v(x_j)\|^2 = \sum_{j=1}^{m} \|x_j\|^2 - \min_{0 \neq v \in \mathbb{R}^m} \sum_{j=1}^{m} \|P_v^\perp(x_j)\|^2
\end{aligned}
$$

where $P_v(x)$ $(P_v^\perp(x))$ is the projection of $x$ onto the space spanned by $v$ (space perpendicular to $v$), since $\|x\|^2 = \|P_v(x)\|^2 + \|P_v^\perp(x)\|^2$. It follows that the first eigenvector is characterised as the direction for which sum of the squares of the residuals is minimal.

Applying the same line of reasoning to the first equality of Theorem 3, delivers the following equality

$$
\lambda_k = \max_{\dim(V)=k} \min_{0 \neq v \in V} \sum_{j=1}^{m} \|P_v(x_j)\|^2. \tag{1}
$$

Notice that this characterisation implies that if $v^k$ is the $k$-th eigenvector of $N$, then

$$
\lambda_k = \sum_{j=1}^{m} \|P_{v^k}(x_j)\|^2, \tag{2}
$$

which in turn implies that if $V_k$ is the space spanned by the first $k$ eigenvectors, then

$$
\sum_{i=1}^{k} \lambda_i = \sum_{j=1}^{m} \|P_{V_k}(x_j)\|^2 = \sum_{j=1}^{m} \|x_j\|^2 - \sum_{j=1}^{m} \|P_{V_k}^\perp(x_j)\|^2, \tag{3}
$$

where $P_V(x)$ $(P_V^\perp(x))$ is the projection of $x$ into the space $V$ (space perpendicular to $V$). It readily follows by induction over the dimension of $V$ that we can equally characterise the sum of the first $k$ and last $m - k$ eigenvalues by

$$
\begin{aligned}
\sum_{i=1}^{k} \lambda_i &= \max_{\dim(V)=k} \sum_{j=1}^{m} \|P_V(x_j)\|^2 = \sum_{j=1}^{m} \|x_j\|^2 - \min_{\dim(V)=k} \sum_{j=1}^{m} \|P_V^\perp(x_j)\|^2, \\
\sum_{i=k+1}^{m} \lambda_i &= \sum_{j=1}^{m} \|x_j\|^2 - \sum_{i=1}^{k} \lambda_i = \min_{\dim(V)=k} \sum_{j=1}^{m} \|P_V^\perp(x_j)\|^2. \tag{4}
\end{aligned}
$$

Hence, as for the case when $k = 1$, the subspace spanned by the first $k$ eigenvalues is characterised as that for which the sum of the squares of the residuals is minimal. Frequently, we consider all of the above as occurring in a kernel defined feature space, so that wherever we have written $x_j$ we should have put $\phi(x_j)$, where $\phi$ is the corresponding projection.

## 3 Concentration of eigenvalues

The previous section outlined the relatively well-known perspective that we now apply to obtain the concentration results for the eigenvalues of positive semi-definite

matrices. The key to the results is the characterisation in terms of the sums of residuals given in equations (1) and (4).

**Theorem 4** *Let $K(x, z)$ be a positive semi-definite kernel function on a space $X$, and let $\mu$ be a distribution on $X$. Fix natural numbers $m$ and $1 \leq k < m$ and let $S = (x_1, \ldots, x_m) \in X^m$ be a sample of $m$ points drawn according to $\mu$. Then for all $\epsilon > 0$,*

$$P\{|\frac{1}{m}\lambda_k(S) - \mathbb{E}_S[\frac{1}{m}\lambda_k(S)]| \geq \epsilon\} \leq 2\exp\left(\frac{-2\epsilon^2 m}{R^4}\right),$$

*where $\lambda_k(S)$ is the $k$-th eigenvalue of the matrix $K(S)$ with entries $K(S)_{ij} = K(x_i, x_j)$ and $R^2 = \max_{x \in X} K(x, x)$.*

**Proof**: The result follows from an application of Theorem 1 provided

$$\sup_S |\frac{1}{m}\lambda_k(S) - \frac{1}{m}\lambda_k(S \setminus \{x_i\})| \leq R^2/m.$$

Let $\hat{S} = S \setminus \{x_i\}$ and let $V$ ($\hat{V}$) be the $k$ dimensional subspace spanned by the first $k$ eigenvectors of $K(S)$ ($K(\hat{S})$). Using equation (1) we have

$$
\begin{aligned}
\lambda_k(S) &\geq \min_{v \in \hat{V}} \sum_{j=1}^{m} \|P_v(x_j)\|^2 \geq \min_{v \in \hat{V}} \sum_{j \neq i} \|P_v(x_j)\|^2 = \lambda_k(\hat{S}) \\
\lambda_k(\hat{S}) &\geq \min_{v \in V} \sum_{j \neq i} \|P_v(x_j)\|^2 \geq \min_{v \in V} \sum_{j=1}^{m} \|P_v(x_j)\|^2 - R^2 = \lambda_k(S) - R^2
\end{aligned}
$$

☐

Surprisingly a very similar result holds when we consider the sum of the last $m - k$ eigenvalues.

**Theorem 5** *Let $K(x, z)$ be a positive semi-definite kernel function on a space $X$, and let $\mu$ be a distribution on $X$. Fix natural numbers $m$ and $1 \leq k < m$ and let $S = (x_1, \ldots, x_m) \in X^m$ be a sample of $m$ points drawn according to $\mu$. Then for all $\epsilon > 0$,*

$$P\{|\frac{1}{m}\lambda^{>k}(S) - \mathbb{E}_S[\frac{1}{m}\lambda^{>k}(S)]| \geq \epsilon\} \leq 2\exp\left(\frac{-2\epsilon^2 m}{R^4}\right),$$

*where $\lambda^{>k}(S)$ is the sum of all but the largest $k$ eigenvalues of the matrix $K(S)$ with entries $K(S)_{ij} = K(x_i, x_j)$ and $R^2 = \max_{x \in X} K(x, x)$.*

**Proof**: The result follows from an application of Theorem 1 provided

$$\sup_S |\frac{1}{m}\lambda^{>k}(S) - \frac{1}{m}\lambda^{>k}(S \setminus \{x_i\})| \leq R^2/m.$$

Let $\hat{S} = S \setminus \{x_i\}$ and let $V$ ($\hat{V}$) be the $k$ dimensional subspace spanned by the first $k$ eigenvectors of $K(S)$ ($K(\hat{S})$). Using equation (4) we have

$$
\begin{aligned}
\lambda^{>k}(S) &\leq \sum_{j=1}^{m} \|P_{\hat{V}}^{\perp}(x_j)\|^2 \leq \sum_{j \neq i} \|P_{\hat{V}}^{\perp}(x_j)\|^2 + R^2 = \lambda^{>k}(\hat{S}) + R^2 \\
\lambda^{>k}(\hat{S}) &\leq \sum_{j \neq i} \|P_V^{\perp}(x_j)\|^2 = \sum_{j=1}^{m} \|P_V^{\perp}(x_j)\|^2 - \|P_V^{\perp}(x_i)\|^2 \leq \lambda^{>k}(S)
\end{aligned}
$$

☐

Our next result concerns the concentration of the residuals with respect to a fixed subspace. For a subspace $V$ and training set $S$, we introduce the notation

$$\bar{P}_V(S) = \frac{1}{m} \sum_{i=1}^{m} \|P_V(x_i)\|^2.$$

**Theorem 6** *Let $\mu$ be a distribution on $X$. Fix natural numbers $m$ and a subspace $V$ and let $S = (x_1, \ldots, x_m) \in X^m$ be a sample of $m$ points drawn according to $\mu$. Then for all $\epsilon > 0$,*

$$P\{|\bar{P}_V(S) - \mathbb{E}_S[\bar{P}_V(S)]| \geq \epsilon\} \leq 2\exp\left(\frac{-\epsilon^2 m}{2R^4}\right).$$

**Proof**: The result follows from an application of Theorem 2 provided

$$\sup_{S,\hat{x}_i} |\bar{P}_V(S) - \bar{P}(S \setminus \{x_i\} \cup \{\hat{x}_i\})| \leq R^2/m.$$

Clearly the largest change will occur if one of the points $x_i$ and $\hat{x}_i$ is lies in the subspace $V$ and the other does not. In this case the change will be at most $R^2/m$. □

## 4 Experiments

In order to test the concentration results we performed experiments with the Breast cancer data using a cubic polynomial kernel. The kernel was chosen to ensure that the spectrum did not decay too fast.

We randomly selected 50% of the data as a 'training' set and kept the remaining 50% as a 'test' set. We centered the whole data set so that the origin of the feature space is placed at the centre of gravity of the training set. We then performed an eigenvalue decomposition of the training set. The sum of the eigenvalues greater than the $k$-th gives the sum of the residual squared norms of the training points when we project onto the space spanned by the first $k$ eigenvectors. Dividing this by the average of all the eigenvalues (which measures the average square norm of the training points in the transformed space) gives a fraction residual not captured in the $k$ dimensional projection. This quantity was averaged over 5 random splits and plotted against dimension in Figure 1 as the continuous line. The error bars give one standard deviation. The Figure 1a shows the full spectrum, while Figure 1b shows a zoomed in subwindow. The very tight error bars show clearly the very tight concentration of the sums of tail of eigenvalues as predicted by Theorem 5.

In order to test the concentration results for subsets we measured the residuals of the test points when they are projected into the subspace spanned by the first $k$ eigenvectors generated above for the training set. The dashed lines in Figure 1 show the ratio of the average squares of these residuals to the average squared norm of the test points. We see the two curves tracking each other very closely, indicating that the subspace identified as optimal for the training set is indeed capturing almost the same amount of information in the test points.

## 5 Conclusions

The paper has shown that the eigenvalues of a positive semi-definite matrix generated from a random sample is concentrated. Furthermore the sum of the last $m - k$ eigenvalues is similarly concentrated as is the residual when the data is projected into a fixed subspace.

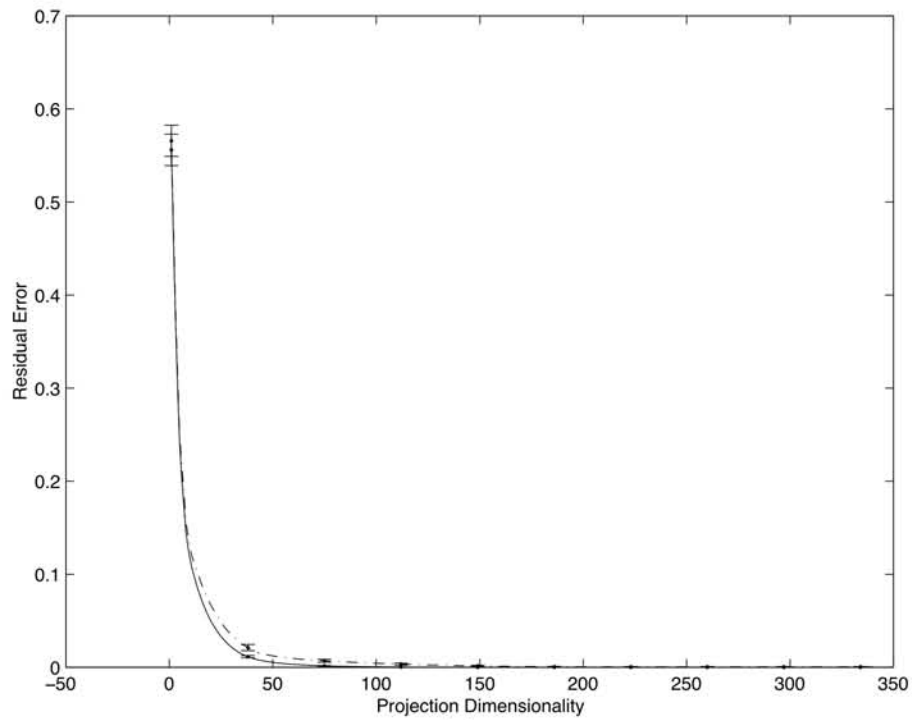

(a)

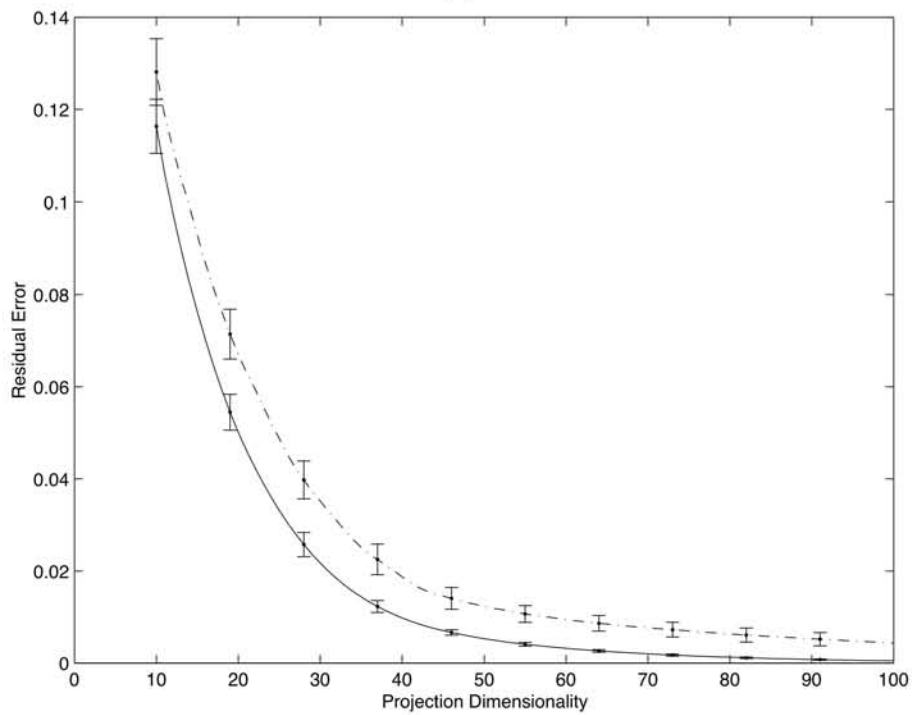

(b)

Figure 1: Plots of the fraction of the average squared norm captured in the subspace spanned by the first $k$ eigenvectors for different values of $k$. Continuous line is fraction for training set, while the dashed line is for the test set. (a) shows the full spectrum, while (b) zooms in on an interesting portion.

Experiments are presented that confirm the theoretical predictions on a real world dataset. The results provide a basis for performing PCA or kernel-PCA from a randomly generated sample, as they confirm that the subset identified by the sample will indeed 'generalise' in the sense that it will capture most of the information in a test sample.

Further research should look at the question of how the space identified by a subsample relates to the eigenspace of the underlying kernel operator.

# References

[1] S. Brin and L. Page. The anatomy of a large-scale hypertextual (web) search engine. In *Proceedings of the Seventh International World Wide Web Conference*, 1998.

[2] Nello Cristianini, Huma Lodhi, and John Shawe-Taylor. Latent semantic kernels for feature selection. Technical Report NC-TR-00-080, NeuroCOLT Working Group, http://www.neurocolt.org, 2000.

[3] J. Kleinberg. Authoritative sources in a hyperlinked environment. In *Proceedings of 9th ACM-SIAM Symposium on Discrete Algorithms*, 1998.

[4] C. McDiarmid. On the method of bounded differences. In *Surveys in Combinatorics 1989*, pages 148–188. Cambridge University Press, 1989.

[5] S. Mika, B. Schölkopf, A. Smola, K.-R. Müller, M. Scholz, and G. Rätsch. Kernel PCA and de-noising in feature spaces. In *Advances in Neural Information Processing Systems 11*, 1998.

[6] Andrew Y. Ng, Alice X. Zheng, and Michael I. Jordan. Link analysis, eigenvectors and stability. In *To appear in the Seventeenth International Joint Conference on Artificial Intelligence (IJCAI-01)*, 2001.
